How Neural Nets Work

Alan Lapedes
Robert Farber
Theoretical Division
Los Alamos National Laboratory
Los Alamos, NM 87545

**Abstract:**

There is presently great interest in the abilities of neural networks to mimic "qualitative reasoning" by manipulating neural incodings of symbols. Less work has been performed on using neural networks to process floating point numbers and it is sometimes stated that neural networks are somehow inherently inaccurate and therefore best suited for "fuzzy" qualitative reasoning. Nevertheless, the potential speed of massively parallel operations make neural net "number crunching" an interesting topic to explore. In this paper we discuss some of our work in which we demonstrate that for certain applications neural networks can achieve significantly higher numerical accuracy than more conventional techniques. In particular, prediction of future values of a chaotic time series can be performed with exceptionally high accuracy. We analyze how a neural net is able to do this , and in the process show that a large class of functions from $R^n \to R^m$ may be accurately approximated by a backpropagation neural net with just two "hidden" layers. The network uses this functional approximation to perform either interpolation (signal processing applications) or extrapolation (symbol processing applications). Neural nets therefore use quite familiar methods to perform their tasks. The geometrical viewpoint advocated here seems to be a useful approach to analyzing neural network operation and relates neural networks to well studied topics in functional approximation.

## 1. Introduction

Although a great deal of interest has been displayed in neural network's capabilities to perform a kind of qualitative reasoning, relatively little work has been done on the ability of neural networks to process floating point numbers in a massively parallel fashion. Clearly, this is an important ability. In this paper we discuss some of our work in this area and show the relation between numerical, and symbolic processing. We will concentrate on the the subject of accurate prediction in a time series. Accurate prediction has applications in many areas of signal processing. It is also a useful, and fascinating ability, when dealing with natural, physical systems. Given some data from the past history of a system, can one accurately predict what it will do in the future?

Many conventional signal processing tests, such as correlation function analysis, cannot distinguish deterministic chaotic behavior from from stochastic noise. Particularly difficult systems to predict are those that are nonlinear and chaotic. Chaos has a technical definition based on nonlinear, dynamical systems theory, but intuitivly means that the system is deterministic but "random," in a rather similar manner to deterministic, pseudo random number generators used on conventional computers. Examples of chaotic systems in nature include turbulence in fluids (D. Ruelle, 1971; H. Swinney, 1978), chemical reactions (K. Tomita, 1979), lasers (H. Haken, 1975), plasma physics (D. Russel, 1980) to name but a few. Typically, chaotic systems also display the full range of nonlinear behavior (fixed points, limit cycles etc.) when parameters are varied, and therefore provide a good testbed in which to investigate techniques of nonlinear signal processing. Clearly, if one can uncover the underlying, deterministic algorithm from a chaotic time series, then one may be able to predict the future time series quite accurately.

In this paper we review and extend our work (Lapedes and Farber,1987) on predicting the behavior of a particular dynamical system, the Glass-Mackey equation. We feel that the method will be fairly general, and use the Glass-Mackey equation solely for illustrative purposes. The Glass-Mackey equation has a strange attractor with fractal dimension controlled by a constant parameter appearing in the differential equation. We present results on a neural network's ability to predict this system at two values of this parameter, one value corresponding to the onset of chaos, and the other value deeply in the chaotic regime. We also present the results of more conventional predictive methods and show that a neural net is able to achieve significantly better numerical accuracy. This particular system was chosen because of D. Farmer's and J. Sidorowich's (D. Farmer, J. Sidorowich, 1987) use of it in developing a new, non-neural net method for predicting chaos. The accuracy of this non-neural net method, and the neural net method, are roughly equivalent, with various advantages or disadvantages accruing to one method or the other depending on one's point of view. We are happy to acknowledge many valuable discussions with Farmer and Sidorowich that has led to further improvements in each method.

We also show that a neural net never needs more than two hidden layers to solve most problems . This statement arises from a more general argument that a neural net can approximate functions from $R^n \rightarrow R^m$ with only two hidden layers, and that the accuracy of the approximation is controlled by the number of neurons in each layer. The argument assumes that the global minimum to the backpropagation minimization problem may be found, or that a local minima very close in value to the global minimum may be found. This seems to be the case in the examples we considered, and in many examples considered by other researchers, but is never guaranteed. The conclusion of an upper bound of two hidden layers is related to a similar conclusion of R. Lipman (R. Lipman, 1987) who has previously analyzed the number of hidden layers needed to form arbitrary decision regions for symbolic processing problems. Related issues are discussed by J. Denker (J. Denker et.al. 1987) It is easy to extend the argument to draw similar conclusions about an upper bound of two hidden layers for symbol processing and to place signal processing, and symbol processing in a common theoretical framework.

## 2. Backpropagation

Backpropagation is a learning algorithm for neural networks that seeks to find weights, $T_{ij}$, such that given an input pattern from a training set of pairs of Input/Output patterns, the network will produce the Output of the training set given the Input. Having learned this mapping between I and O for the training set, one then applies a new, previously unseen Input, and takes the Output as the "conclusion" drawn by the neural net based on having learned fundamental relationships between Input and Output from the training set. A popular configuration for backpropagation is a totally feedforward net (Figure 1) where Input feeds up through "hidden layers" to an Output layer.

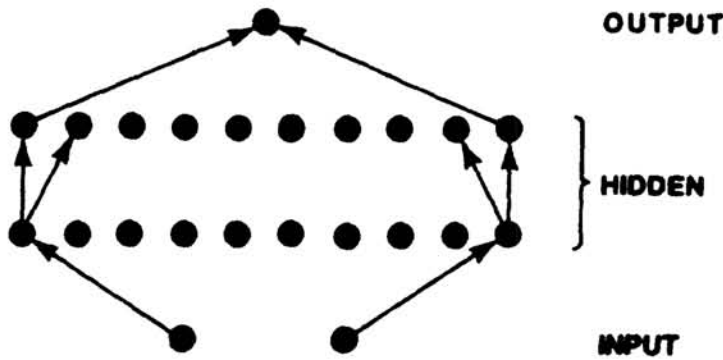

**OUTPUT**

**HIDDEN**

**INPUT**

Figure 1.
A feedforward neural
net. Arrows schemat-
ically indicate full
feedforward connect-
ivity

Each neuron forms a weighted sum of the inputs from previous layers to which it is connected, adds a threshold value, and produces a nonlinear function of this sum as its output value. This output value serves as input to the future layers to which the neuron is connected, and the process is repeated. Ultimately a value is produced for the outputs of the neurons in the Output layer. Thus, each neuron performs:

$$X_i^{out} = g\left(\sum_j T_{ij}X_j^{in} + \theta_i\right) \tag{1}$$

where $T_{ij}$ are continuous valued, positive or negative weights, $\theta_i$ is a constant, and g(x) is a nonlinear function that is often chosen to be of a sigmoidal form. For example, one may choose

$$g(x) = \frac{1}{2}\left(1 + tanhx\right) \tag{2}$$

where tanh is the hyperbolic tangent, although the exact formula of the sigmoid is irrelevant to the results.
If $t_i^{(p)}$ are the target output values for the $p^{th}$ Input pattern then ones trains the network by minimizing

$$E = \sum_p \sum_i \left(t_i^{(p)} - 0_i^{(p)}\right)^2 \tag{3}$$

where $t_i^{(p)}$ is the target output values (taken from the training set) and $0_i^{(p)}$ is the output of the network when the $p^{th}$ Input pattern of the training set is presented on the Input layer. i indexes the number of neurons in the Output layer.

An iterative procedure is used to minimize E. For example, the commonly used steepest descents procedure is implemented by changing $T_{ij}$ and $\theta_i$ by $\Delta T_{ij}$ and $\Delta \theta_i$ where

$$\Delta T_{ij} = -\frac{\partial E}{\partial T_{ij}} \cdot \epsilon \tag{4a}$$

$$\Delta \theta_i = -\frac{\partial E}{\partial \theta_i} \cdot \epsilon \tag{4b}$$

This implies that $\Delta E < 0$ and hence E will decrease to a local minimum. Use of the chain rule and definition of some intermediate quantities allows the following expressions for $\Delta T_{ij}$ to be obtained (Rumelhart, 1987):

$$\Delta T_{ij} = \sum_p \epsilon \delta_i^{(p)} 0_j^{(p)} \tag{5a}$$

$$\Delta \theta_i = \epsilon \sum_{\wedge} \delta_i^{(p)} \tag{5b}$$

where

$$\delta_i^{(p)} = \left( t_i^{(p)} - 0_i^{(p)} \right) 0_i^{(p)} (1 - 0_i^{(p)}) \tag{6}$$

if i is labeling a neuron in the Output layer; and

$$\delta_i^{(p)} = 0_i^{(p)} (1 - 0_i^{(p)}) \sum_j T_{ij} \delta_j^{(p)} \tag{7}$$

if i labels a neuron in the hidden layers. Therefore one computes $\delta_i^{(p)}$ for the Output layer first, then uses Eqn. (7) to computer $\delta_i^{(p)}$ for the hidden layers, and finally uses Eqn. (5) to make an adjustment to the weights. We remark that the steepest descents procedure in common use is extremely slow in simulation, and that a better minimization procedure, such as the classic conjugate gradient procedure (W. Press, 1986), can offer quite significant speedups. Many applications use bit representations (0,1) for symbols, and attempt to have a neural net learn fundamental relationships between the symbols. This procedure has been successfully used in converting text to speech (T. Sejnowski, 1986) and in determining whether a given fragment of DNA codes for a protein or not (A. Lapedes, R. Farber, 1987).

There is no fundamental reason, however, to use integer's as values for Input and Output. If the Inputs and Outputs are instead a collection of floating point numbers, then the network, after training, yields a specific continuous function in n variables (for n inputs) involving $g(x)$ (i.e. hyperbolic tanh's) that provides a type of nonlinear, least mean square interpolant formula for the discrete set of data points in the training set. Use of this formula $0 = f(I_1, I_2, \ldots I_n)$ when given a new input not in the training set, is then either interpolation or extrapolation.

Since the Output values, when assumed to be floating point numbers may have a dynamic range great than $[0,1]$, one may modify the $g(x)$ on the Output layer to be a linear function, instead of sigmoidal, so as to encompass the larger dynamic range. Dynamic range of the Input values is not so critical, however we have found that numerical problems may be avoided by scaling the Inputs (and

also the Outputs) to [0,1], training the network, and then rescaling the $T_{ij}$, $\theta_i$ to encompass the original dynamic range. The point is that scale changes in I and O may, for feedforward networks, always be absorbed in the $T_{ij}$, $\theta_i$ and vice versa. We use this procedure (backpropagation, conjugate gradient, linear outputs and scaling) in the following section to predict points in a chaotic time series.

## 3. Prediction

Let us consider situations in Nature where a system is described by nonlinear differential equations. This is faily generic. We choose a particular nonlinear equation that has an infinite dimensional phase space, so that it is similar to other infinite dimensional systems such as partial differential equations. A differential equation with an infinite dimensional phase space (i.e. an infinite number of values are necessary to describe the initial condition) is a delay, differential equation. We choose to consider the time series generated by the Glass-Mackey equation:

$$\dot{x} = \frac{ax(t-\tau)}{1 + x^{10}(t-\tau)} - bx(t) \qquad (8)$$

This is a nonlinear differential, delay equation with an initial condition specified by an initial function defined over a strip of width $\tau$ (hence the infinite dimensional phase space i.e. initial functions, not initial constants are required). Choosing this function to be a constant function, and a $= .2$, b $= .1$, and $\tau = 17$ yields a time series, x(t), (obtained by integrating Eqn. (8)), that is chaotic with a fractal attractor of dimension 2.1. Increasing $\tau$ to 30 yields more complicated evolution and a fractal dimension of 3.5. The time series for 500 time steps for $\tau=30$ (time in units of $\tau$) is plotted in Figure 2. The nonlinear evolution of the system collapses the infinite dimensional phase space down to a low (approximately 2 or 3 dimensional) fractal, attracting set. Similar chaotic systems are not uncommon in Nature.

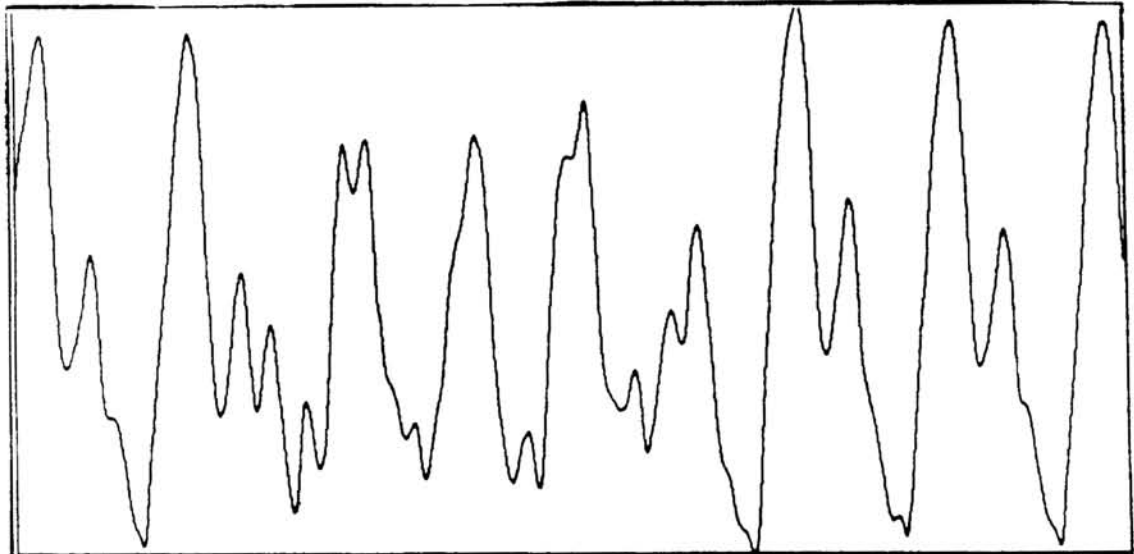

Figure 2. Example time series at tau = 30.

The goal is to take a set of values of x() at discrete times in some time window containing times less than t, and use the values to accurately predict x(t + P), where P is some prediction time step into the future. One may fix P, collect statistics on accuracy for many prediction times t (by sliding the window along the time series), and then increase P and again collect statistics on accuracy. This one may observe how an average index of accuracy changes as P is increased. In terms of Figure 2 we will select various prediction time steps, P, that correspond to attempting to predict within a "bump," to predicting a couple of "bumps" ahead. The fundamental nature of chaos dictates that prediction accuracy will decrease as P is increased. This is due to inescapable inaccuracies of finite precision in specifying the x(t) at discrete times in the past that are used for predicting the future. Thus, all predictive methods will degrade as P is increased – the question is "How rapidly does the error increase with P?" We will demonstrate that the neural net method can be orders of magnitude more accurate than conventional methods at large prediction time steps, P.

Our goal is to use backpropagation, and a neural net, to construct a function

$$0(t + P) = f\left(I_1(t), I_2(t - \Delta) \ldots I_m(t - m\Delta)\right) \qquad (9)$$

where $0(t + P)$ is the output of a single neuron in the Output layer, and $I_1 \to I_m$ are input neurons that take on values $x(t), x(t - \Delta) \ldots x(t - m\Delta)$, where $\Delta$ is a time delay. $0(t + P)$ takes on the value x(t + P). We chose the network configuation of Figure 1.

We construct a training set by selecting a set of input values:

$$I_1 = x(t_p)$$

$$I_2 = x(t_p - \Delta) \qquad (10)$$

$$I_m = x(t_p - m\Delta)$$

with associated output values $0 = x(t_p + P)$, for a collection of discrete times that are labelled by $t_p$. Typically we used 500 I/O pairs in the training set so that p ranged from $1 \to 500$. Thus we have a collection of 500 sets of $\{I_1^{(p)}, I_2^{(p)}, \ldots, I_m^{(p)}; 0^{(p)}\}$ to use in training the neural net. This procedure of using delayed sampled values of x(t) can be implemented by using tapped delay lines, just as is normally done in linear signal processing applications, (B. Widrow, 1985). Our prediction procedure is a straightforward nonlinear extension of the linear Widrow Hoff algorithm. After training is completed, prediction is performed on a new set of times, $t_p$, not in the training set i.e. for p = 500.

We have not yet specified what m or $\Delta$ should be, nor given any indication why a formula like Eqn. (9) should work at all. An important theorem of Takens (Takens, 1981) states that for flows evolving to compact attracting manifolds of dimension $d_A$, that a functional relation like Eqn. (9) does exist, and that m lies in the range $d_A < m + 1 < 2d_A + 1$. We therefore choose m = 4, for $\tau = 30$. Takens provides no information on $\Delta$ and we chose $\Delta = 6$ for both cases. We found that a few different choices of m and $\Delta$ can affect accuracy by a factor of 2 - a somewhat significant but not overwhelming sensitivity, in view of the fact that neural nets tend to be orders of magnitude more accurate than other methods. Takens theorem gives no information on the form of f() in Eqn. (9). It therefore

is necessary to show that neural nets provide a robust approximating procedure for continuous f(), which we do in the following section. It is interesting to note that attempts to predict future values of a time series using past values of x(t) from a tapped delay line is a common procedure in signal processing, and yet there is little, if any, reference to results of nonlinear dynamical systems theory showing why any such attempt is reasonable.

After training the neural net as described above, we used it to predict 500 new values of x(t) in the future and computed the average accuracy for these points. The accuracy is defined to be the average root mean square error, divided by a constant scale factor, which we took to be the standard deviation of the data. It is necessary to remove the scale dependence of the data and dividing by the standard deviation of the data provides a scale to use. Thus the resulting "index of accuracy" is insensitive to the dynamic range of x(t).

As just described, if one wanted to use a neural net to continuously predict x(t) values at, say, 6 time steps past the last observed value (i.e. wanted to construct a net predicting x(t + 6)) then one would train one network, at P = 6, to do this. If one wanted to always predict 12 time steps past the last observed x(t) then a separate, P = 12, net would have to be trained. We, in fact, trained separate networks for P ranging between 6 and 100 in steps of 6. The index of accuracy for these networks (as obtained by computing the index of accuracy in the prediction phase) is plotted as curve D in Figure 3. There is however an alternate way to predict. If one wished to predict, say, x(t + 12) using a P = 6 net, then one can iterate the P = 6 net. That is, one uses the P = 6 net to predict the x(t +6) values, and then feeds x(t +6) back into the input line to predict x(t + 12) using the **predicted** x(t + 6) value instead of the **observed** x(t + 6) value. In fact, one can't use the observed x(t +6) value, because it hasn't been observed yet – the rule of the game is to use only data occurring at time t and before, to predict x(t +12). This procedure corresponds to iterating the map given by Eqn. (9) to perform prediction at multiples of P. Of course, the delays, $\Delta$, must be chosen commensurate with P.

This iterative method of prediction has potential dangers. Because (in our example of iterating the P = 6 map) the predicted x(t + 6) is always made with some error, then this error is compounded in iteration, because predicted, and not observed values, are used on the input lines. However, one may predict more accurately for smaller P, so it may be the case that choosing a very accurate small P prediction, and iterating, can ultimately achieve higher accuracy at the larger P's of interest. This turns out to be true, and the iterated net method is plotted as curve E in Figure 3. It is the best procedure to use. Curves A,B,C are alternative methods (iterated polynomial,Widrow-Hoff, and non-iterated polynomial respectively. More information on these conventional methods is in (Lapedes and Farber, 1987) ).

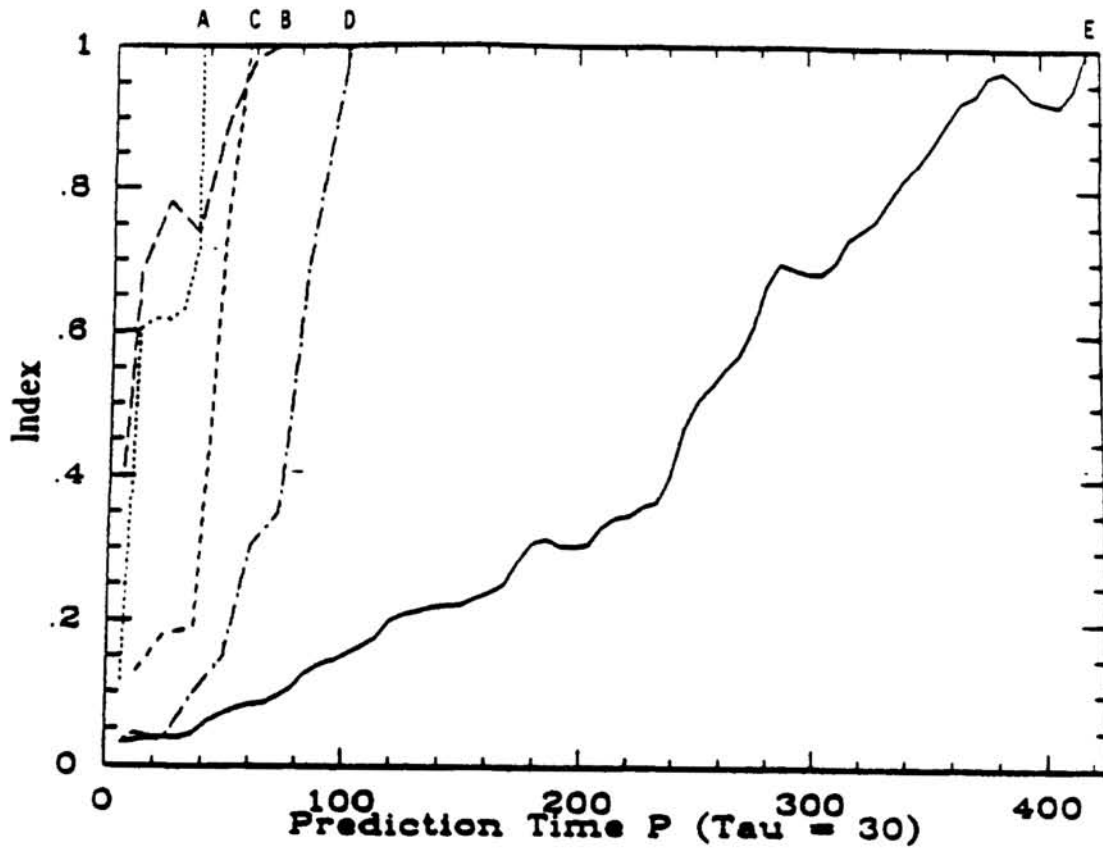

Figure 3.

## 4. Why It Works

Consider writing out explicitly Eqn. (9) for a two hidden layer network where the output is assumed to be a linear neuron. We consider Input connects to Hidden Layer 1, Hidden Layer 1 to Hidden Layer 2, and Hidden Layer 2 to Output. Therefore:

$$0_\ell = \sum_{k \epsilon H_2} T_{\ell k} g \left( \sum_{i \epsilon H_1} T_{ki} g \left( \sum_{j \epsilon I} T_{ij} I_j + \theta_j \right) + \theta_k \right) + \theta_\ell \qquad (11)$$

Recall that the output neurons a linear computing element so that only two g()s occur in formula (11), due to the two nonlinear hidden layers. For ease in later analysis, let us rewrite this formula as

$$0_\ell = \sum_{k \epsilon H_2} T_{\ell k} g \left( SUM_k + \theta_k \right) + \theta_\ell \qquad (12a)$$

where

$$SUM_k = \sum_{i \epsilon H_1} T_{ki} g \left( \sum_{j \epsilon I} T_{ij} I_j + \theta_j \right). \qquad (12b)$$

The T's and $\theta$'s are specific numbers specified by the training algorithm, so that after training is finished one has a relatively complicated formula (12a, 12b) that expresses the Output value as a specific, known, function of the Input values:

$$0_\ell = f(I_1, I_2, \ldots I_m).$$

A functional relation of this form, when there is only one output, may be viewed as surface in m + 1 dimensional space, in exactly the same manner one interprets the formula z = f(x,y) as a two dimensional surface in three dimensional space. The general structure of f() as determined by Eqn. (12a, 12b) is in fact quite simple. From Eqn. (12b) we see that one first forms a sum of g() functions (where g() is s sigmoidal function) and then from Eqn. (12a) one forms yet another sum involving g() functions. It may at first be thought that this special, simple form of f() restricts the type of surface that may be represented by $0_\ell = f(I_j)$. This initial thought is wrong – the special form of Eqn. (12) is actually a general representation for quite arbitrary surfaces.

To prove that Eqn. (12) is a reasonable representation for surfaces we first point out that surfaces may be approximated by adding up a series of "bumps" that are appropriately placed. An example of this occurs in familiar Fourier analysis, where wave trains of suitable frequency and amplitude are added together to approximate curves (or surfaces). Each half period of each wave of fixed wavelength is a "bump," and one adds all the bumps together to form the approximant. Let us now see how Eqn. (12) may be interpreted as adding together bumps of specified heights and positions. First consider $SUM_k$ which is a sum of g( ) functions. In Figure (4) we plot an example of such a g() function for the case of two inputs.

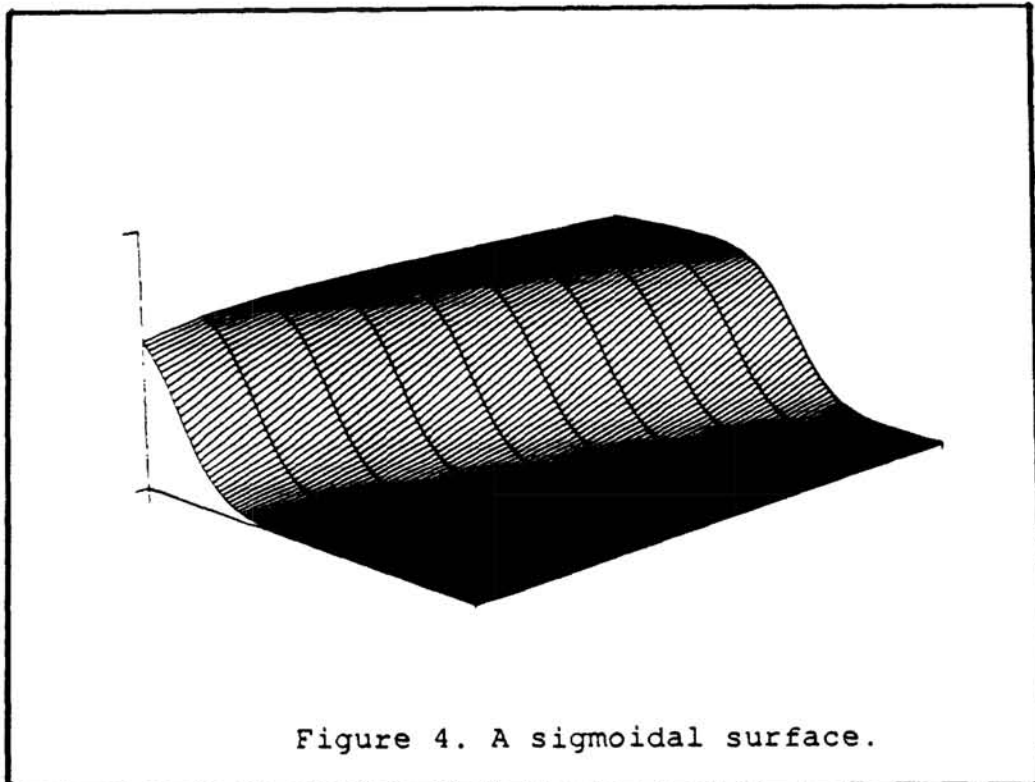

Figure 4. A sigmoidal surface.

The orientation of this sigmoidal surface is determined by $T_{ij}$, the position by $\theta_j$, and height by $T_{ki}$. Now consider another g() function that occurs in SUM$_k$. The $\theta_j$ of the second g() function is chosen to displace it from the first, the $T_{ij}$ is chosen so that it has the same orientation as the first, and $T_{ki}$ is chosen to have opposite sign to the first. These two g( ) functions occur in SUM$_k$, and so to determine their contribution to SUM$_k$ we sum them together and plot the result in Figure (5). The result is a ridged surface.

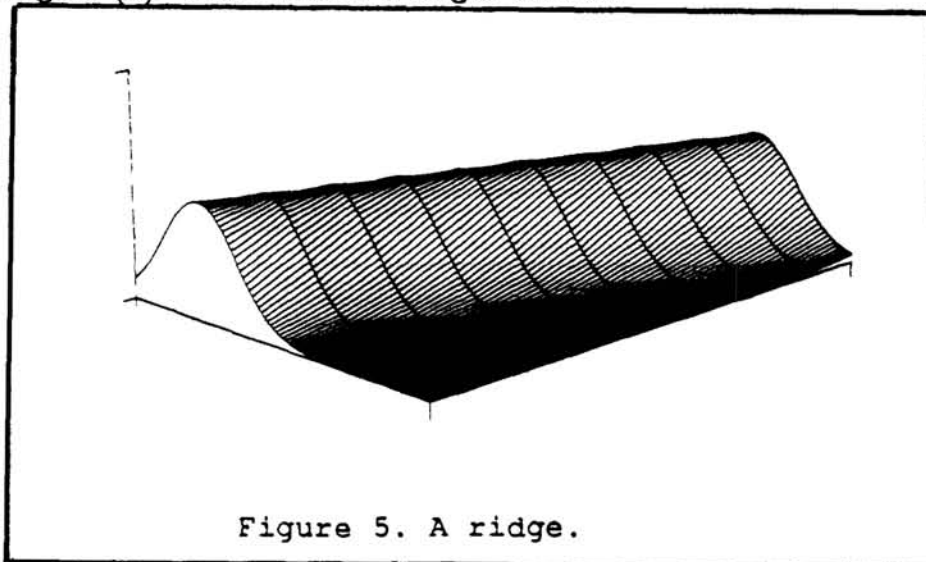

Figure 5. A ridge.

Since our goal is to obtain localized bumps we select another pair of g() functions in SUM$_k$, add them together to get a ridged surface perpendicular to the first ridged surface, and then add the two perpendicular ridged surfaces together to see the contribution to SUM$_k$. The result is plotted in Figure (6).

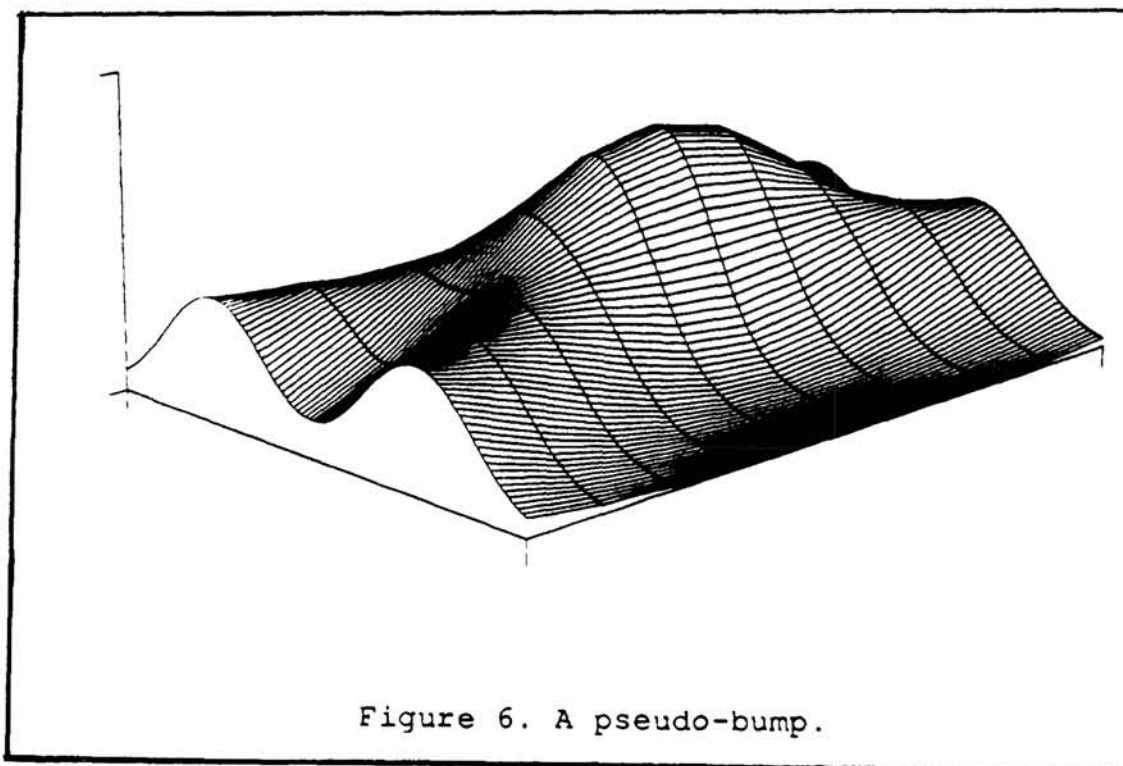

Figure 6. A pseudo-bump.

We see that this almost worked, in so much as one obtains a local maxima by this procedure. However there are also saddle-like configurations at the corners which corrupt the bump we were trying to obtain. Note that one way to fix this is to take $g(\text{SUM}_k + \theta_k)$ which will, if $\theta_k$ is chosen appropriately, depress the local minima and saddles to zero while simultaneously sending the central maximum towards 1. The result is plotted in Figure (7) and is the sought after bump.

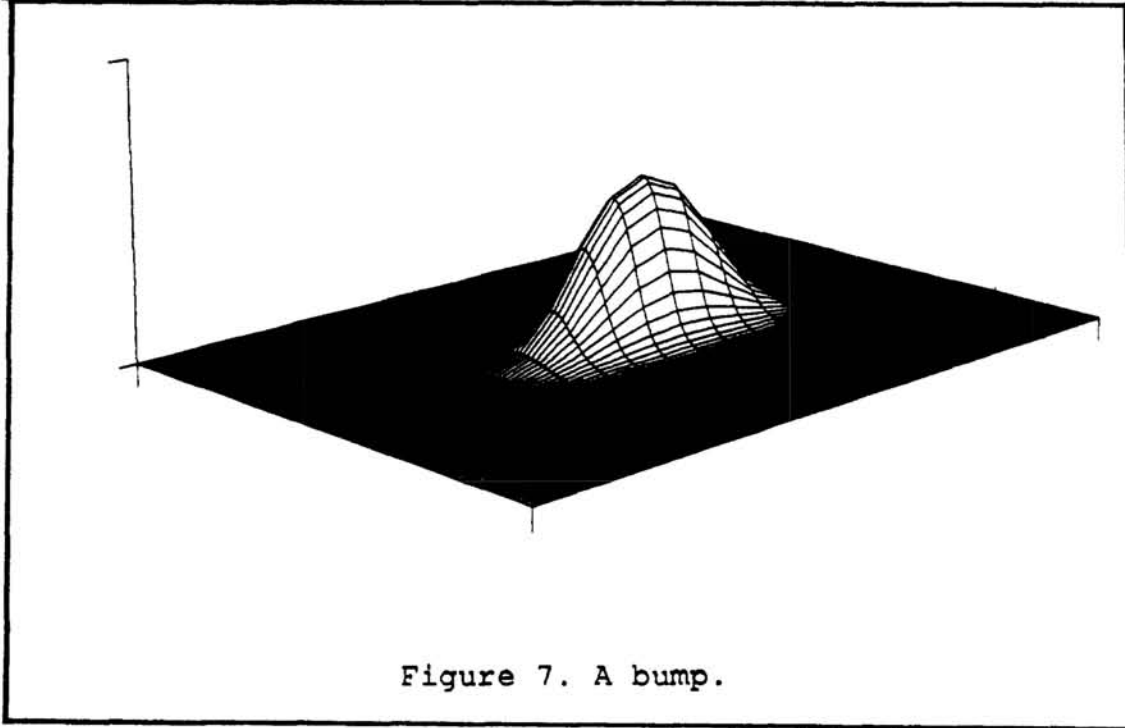

Figure 7. A bump.

Furthermore, note that the necessary g() function is supplied by Eqn. (12). Therefore Eqn. (12) is a procedure to obtain localized bumps of arbitrary height and position. For two inputs, the $k^{th}$ bump is obtained by using four g() functions from $\text{SUM}_k$ (two g() functions for each ridged surface and two ridged surfaces per bump) and then taking g() of the result in Eqn. (12a). The height of the $k^{th}$ bump is determined by $T_{\rho k}$ in Eqn. (12a) and the k bumps are added together by that equation as well. The general network architecture which corresponds to the above procedure of adding two g() functions together to form a ridge, two perpendicular ridges together to form a pseudo-bump, and the final g() to form the final bump is represented in Figure (8). To obtain any number of bumps one adds more neurons to the hidden layers by repeatedly using the connectivity of Figure (8) as a template (i.e. four neurons per bump in Hidden Layer 1, and one neuron per bump in Hidden Layer 2).

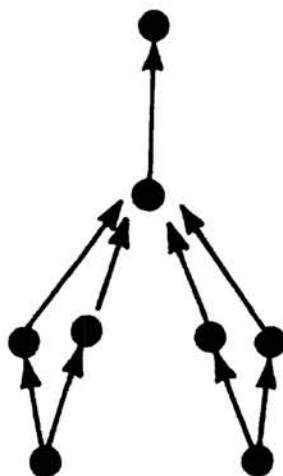

Figure 8. Connectivity needed to obtain one bump. Add four more neurons to Hidden layer 1, and one more neuron to Hidden Layer 2, for each additional bump.

One never needs more than two layers, or any other type of connectivity than that already schematically specified by Figure (8). The accuracy of the approximation depends on the number of bumps, which in turn is specified, by the number of neurons per layer. This result is easily generalized to higher dimensions (more than two Inputs) where one needs 2m hiddens in the first hidden layer, and one hidden neuron in the second layer for each bump.

The argument given above also extends to the situation where one is processing symbolic information with a neural net. In this situation, the Input information is coded into bits (say 0s and 1s) and similarly for the Output. Or, the Inputs may still be real valued numbers, in which case the binary output is attempting to group the real valued Inputs into separate classes. To make the Output values tend toward 0 and 1 one takes a third and final g() on the output layer, i.e. each output neuron is represented by $g(O_\ell)$ where $O_\ell$ is given in Eqn. (11). Recall that up until now we have used linear neurons on the output layer. In typical backpropagation examples, one never actually achieves a hard 0 or 1 on the output layers but achieves instead some value between 0.0 and 1.0. Then typically any value over 0.5 is called 1, and values under 0.5 are called 0. This "postprocessing" step is not really outside the framework of the network formalism, because it may be performed by merely increasing the slope of the sigmoidal function on the Output layer. Therefore the only effect of the third and final g() function used on the Output layer in symbolic information processing is to pass a hyperplane through the surface we have just been discussing. This plane cuts the surface, forming "decision regions," in which high values are called 1 and low values are called 0. Thus we see that the heart of the problem is to be able to form surfaces in a general manner, which is then cut by a hyperplane into general decision regions. We are therefore able to conclude that the network architecture consisting of just two hidden layers is sufficient for learning any symbol processing training set. For Boolean symbol mappings one need not use the second hidden layer to remove the saddles on the bump (c.f. Fig. 6). The saddles are lower than the central maximum so one may choose a threshold on the output layer to cut the bump at a point over the saddles to yield the correct decision region. Whether this representation is a reasonable one for subsequently achieving good prediction on a prediction set, as opposed to "memorizing" a training set, is an issue that we address below.

We also note that use of Sigma II; units (Rummelhart, 1986) or high order correlation nets (Y.-C. Lee, 1987) is an attempt to construct a surface by a general polynomial expansion, which is then cut by a hyperplane into decision regions, as in the above. Therefore the essential element of all these neural net learning algorithms are identical (i.e. surface construction), only the particular method of parameterizing the surface varies from one algorithm to another. This geometrical viewpoint, which provides a unifying framework for many neural net algorithms, may provide a useful framework in which to attempt construction of new algorithms.

Adding together bumps to approximate surfaces is a reasonable procedure to use when dealing with real valued inputs. It ties in to general approximation theory (c.f. Fourier series, or better yet, B splines), and can be quite successful as we have seen. Clearly some economy is gained by giving the neural net bumps to start with, instead of having the neural net form its own bumps from sigmoids. One way to do this would be to use multidimensional Gaussian functions with adjustable parameters.

The situation is somewhat different when processing symbolic (binary valued) data. When input symbols are encoded into N bit bit-strings then one has well defined input values in an N dimensional input space. As shown above, one can learn the training set of input patterns by appropriately forming and placing bump surfaces over this space. This is an effective method for **memorizing** the training set, but a very poor method for obtaining correct predictions on new input data. The point is that, in contrast to real valued inputs that come from, say, a chaotic time series, the input points in symbolic processing problems are widely separated and the bumps do not add together to form smooth surfaces. Furthermore, each input bit string is a corner of an $2^N$ vertex hypercube, and there is no sense in which one corner of a hypercube is surrounded by the other corners. Thus the commonly used input representation for symbolic processing problems requires that the neural net **extrapolate** the surface to make a new prediction for a new input pattern (i.e. new corner of the hypercube) and not interpolate, as is commonly the case for real valued inputs. Extrapolation is a farmore dangerous procedure than interpolation, and in view of the separated bumps of the training set one might expect on the basis of this argument that neural nets would fail dismally at symbol processing. This is not the case.

The solution to this apparent conundrum, of course, is that although it is sufficient for a neural net to learn a symbol processing training set by forming bumps it is not necessary for it to operate in this manner. The simplest example of this occurs in the XOR problem. One can implement the input/output mapping for this problem by duplicating the hidden layer architecture of Figure (8) appropiately for two bumps ( i.e. 8 hiddens in layer 1, 2 hiddens in layer 2). As discussed above, for Boolean mappings, one can even eliminate the second hidden layer. However the architecture of Figure (9) will also suffice.

Figure 9. Connectivity for XOR

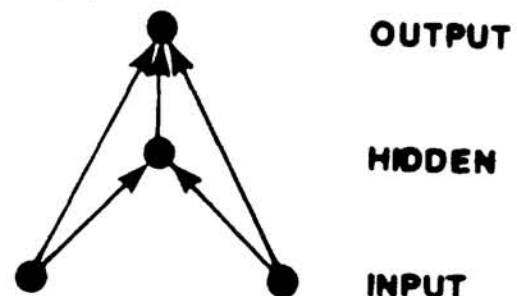

Plotting the output of this network, Figure(9), as a function of the two inputs yields a ridge orientated to run between (0,1) and (1,0) Figure(10). Thus a neural net may learn a symbolic training set without using bumps, and a high dimensional version of this process takes place in more complex symbol processing tasks.Ridge/ravine representations of the training data are considerably more efficient than bumps (less hidden neurons and weights) and the extended nature of the surface allows reasonable predictions i.e. extrapolations.

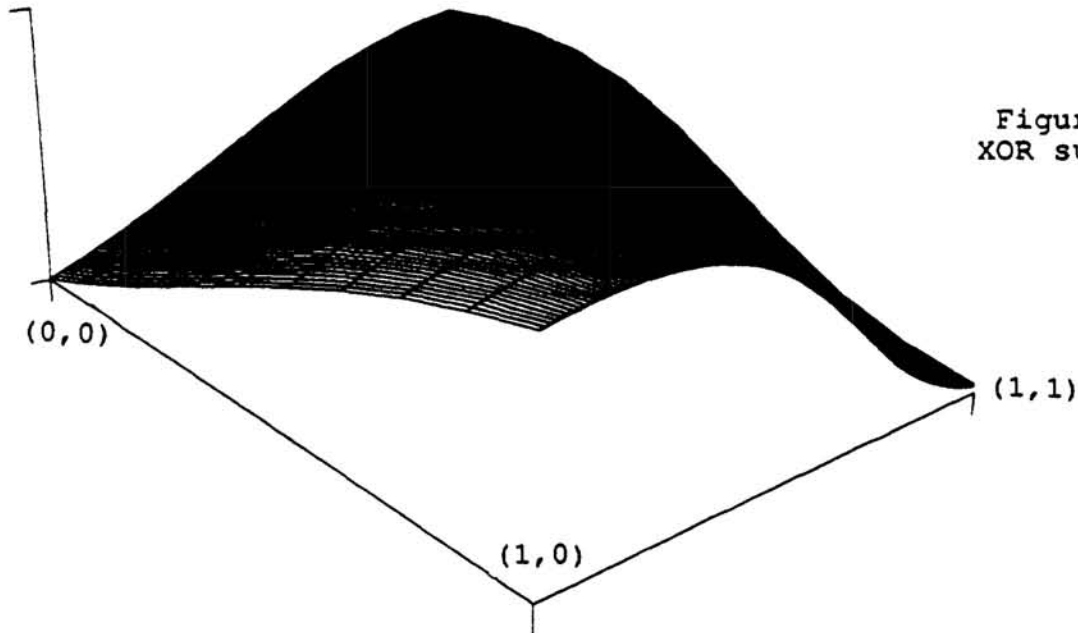

Figure 10
XOR surface

## 5. Conclusions

Neural nets, in contrast to popular misconception, are capable of quite accurate number crunching, with an accuracy for the prediction problem we considered that exceeds conventional methods by orders of magnitude. Neural nets work by constructing surfaces in a high dimensional space, and their operation when performing signal processing tasks on real valued inputs, is closely related to standard methods of functional approximation. One does not need more than two hidden layers for processing real valued input data, and the accuracy of the approximation is controlled by the number of neurons per layer, and not the number of layers. We emphasize that although two layers of hidden neurons are sufficient they may not be efficient. Multilayer architectures may provide very efficient networks (in the sense of number of neurons and number of weights) that can perform accurately and with minimal cost.

Effective prediction for symbolic input data is achieved by a slightly different method than that used for real value inputs. Instead of forming localized bumps (which would accurately represent the training data but would not predict well on new inputs) the network can use ridge/ravine like surfaces (and generalizations thereof) to efficiently represent the scattered input data. While neural nets generally perform prediction by interpolation for real valued data, they must perform extrapolation for symbolic data if the usual bit representations are used. An outstanding problem is why do tanh representations seem to extrapolate well in symbol processing problems? How do other functional bases do? How does the representation for symbolic inputs affect the ability to extrapolate? This geometrical viewpoint provides a unifying framework for examining many neural net algorithms, for suggesting questions about neural net operation, and for relating current neural net approaches to conventional methods.

**Acknowledgments**

We thank Y. C. Lee, J. D. Farmer, and J. Sidorovich for a number of valuable discussions.

**References**

C. Barnes, C. Burks, R. Farber, A. Lapedes, K. Sirotkin, "Pattern Recognition by Neural Nets in Genetic Databases", manuscript in preparation

J. Denker et. al.,"Automatic Learning, Rule Extraction,and Generalization", ATT, Bell Laboratories preprint, 1987

D. Farmer, J.Sidorowich, Phys.Rev. Lett., 59(8), p. 845,1987

H. Haken, Phys. Lett. A53, p77 (1975)

A. Lapedes, R. Farber "Nonlinear Signal Processing Using Neural Networks: Prediction and System Modelling", LA-UR87-2662,1987

Y.C. Lee, Physica 22D,(1986)

R. Lippman, IEEE ASAP magazine,p.4, 1987

D. Ruelle, F. Takens, Comm. Math. Phys. 20, p167 (1971)

D. Rummelhart, J. McClelland in "Parallel Distributed Processing" Vol. 1, M.I.T. Press Cambridge, MA (1986)

D. Russel et al., Phys. Rev. Lett. 45, p1175 (1980)

T. Sejnowski et al., "Net Talk: A Parallel Network that Learns to Read Aloud," Johns Hopkins Univ. preprint (1986)

H. Swinney et al., Physics Today 31 (8), p41 (1978)

F. Takens, "Detecting Strange Attractor in Turbulence," Lecture Notes in Mathematics, D. Rand, L. Young (editors), Springer Berlin, p366 (1981)

K. Tomita et al., J. Stat. Phys. 21, p65 (1979)